# A unified model of short-range and long-range motion perception

**Shuang Wu**
Department of Statistics
UCLA
Los Angeles , CA 90095
shuangw@stat.ucla.edu

**Xuming He**
Department of Statistics
UCLA
Los Angeles , CA 90095
hexm@stat.ucla.edu

**Hongjing Lu**
Department of Psychology
UCLA
Los Angeles , CA 90095
hongjing@ucla.edu

**Alan Yuille**
Department of Statistics, Psychology, and Computer Science
UCLA
Los Angeles , CA 90095
yuille@stat.ucla.edu

## Abstract

The human vision system is able to effortlessly perceive both short-range and long-range motion patterns in complex dynamic scenes. Previous work has assumed that two different mechanisms are involved in processing these two types of motion. In this paper, we propose a hierarchical model as a unified framework for modeling both short-range and long-range motion perception. Our model consists of two key components: a data likelihood that proposes multiple motion hypotheses using nonlinear matching, and a hierarchical prior that imposes slowness and spatial smoothness constraints on the motion field at multiple scales. We tested our model on two types of stimuli, random dot kinematograms and multiple-aperture stimuli, both commonly used in human vision research. We demonstrate that the hierarchical model adequately accounts for human performance in psychophysical experiments.

## 1 Introduction

We encounter complex dynamic scenes in everyday life. As illustrated by the motion sequence depicted in Figure 1, humans readily perceive the baseball player's body movements and the faster-moving baseball simultaneously. However, from the computational perspective, this is not a trivial problem to solve. The difficulty is due to the large speed difference between the two objects, i.e, the displacement of the player's body is much smaller than the displacement of the baseball between the two frames. Separate motion systems have been proposed to explain human perception in scenarios like this example. In particular, Braddick [1] proposed that there is a short-range motion system which is responsible for perceiving movements with relatively small displacements (e.g., the player's movement), and a long-range motion system which perceives motion with large displacements (e.g., the flying baseball), which is sometimes called apparent motion. Lu and Sperling [2] have further argued for the existence of three motion systems in human vision. The first and second-order systems conduct motion analysis on luminance and texture information respectively, while the third-order system uses a feature-tracking strategy. In the baseball example, the first-order motion system would be used to perceive the player's movements, but the third-order system would be required for perceiving the faster motion of the baseball. Short-range motion and first-order motion appear to apply to the same class of phenomena, and can be modeled using computational theories that are based on motion energy or related techniques. However, long-range motion and third-order

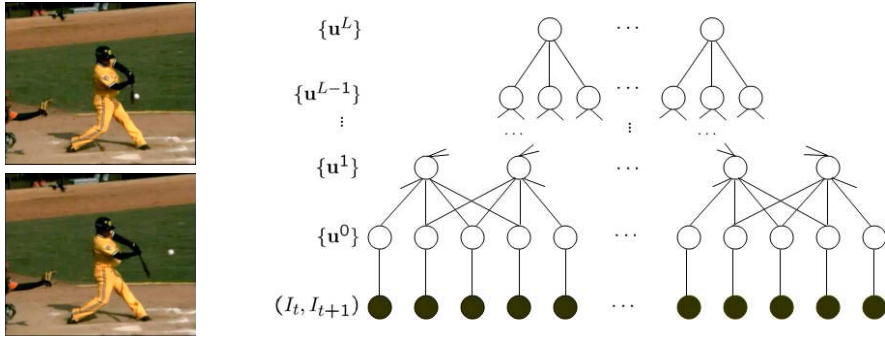

Figure 1: Left panel: Short-range and long-range motion: two frames from a baseball sequence where the ball moves with much faster speed than the other objects. Right panel: A graphical illustration of our hierarchical model in one dimension. Each node represents motion at different location and scales. A child node can have multiple parents, and the prior constraints on motion are expressed by parent-child interactions.

motion employ qualitatively different computational strategies involving tracking features over time, which may require attention-driven processes.

In contrast to these previous multi-system theories [2, 3], we develop a unified single-system framework to account for these phenomena of human motion perception. We model motion estimation as an inference problem which uses flexible prior assumptions about motion flows and statistical models for quantifying the uncertainty in motion measurement. Our model differs from the traditional approaches in two aspects. *First*, the prior model is defined over a hierarchical graph, see Figure 1, where the nodes of the graph represent the motion at different scales. This hierarchical structure is motivated by the human visual system that is organized hierarchically [8, 9, 4]. Such a representation makes it possible to define motion priors and contextual effects at a range of different scales, and so differs from other models of motion perception based on motion priors [5, 6]. This model connects lower level nodes to multiple coarser-level nodes, resulting in a loopy graph structure, which imposes a more flexible prior than tree-structured models (eg. [7]). We define a probability distribution on this graph using potentials defined over the graph cliques to capture spatial smoothness constraints [10] at different scales and slowness constraints [5, 11, 12, 13]. *Second*, our data likelihood terms allow a large space of possible motions, which include both short-range and long-range motion. Locally, the motion is often highly ambiguous (e.g., the likelihood term allows many possible motions) which is resolved in our model by imposing the hierarchical motion prior. Note that we do not coarsen the image and do not rely on coarse-to-fine processing [14]. Instead we use a bottom-up compositional/hierarchical approach where local hypotheses about the motion are combined to form hypotheses for larger regions of the image. This enables us to deal simultaneously with both long-range and short-range motion.

We tested our model using two types of stimuli commonly used in human vision research. The first stimulus type are random dot kinematograms (RDKs), where some of the dots (the signal) move coherently with large displacements, whereas other dots (the noise) move randomly. RDKs are one of the most important stimuli used in both physiological and psychophysical studies of motion perception. For example, electrophysiological studies have used RDKs to analyze the neuronal basis of motion perception, identifying a functional link between the activity of motion-selective neurons and behavioral judgments of motion perception [15]. Psychophysical studies have used RDKs to measure the sensitivity of the human visual system for perceiving coherent motion, and also to infer how motion information is integrated to perceive global motion under different viewing conditions [16]. We used two-frame RDKs as an example of a long-range motion stimulus. The second stimulus type are moving gratings or plaids. These stimuli have been used to study many perceptual phenomena. For example, when randomly orientated lines or grating elements drift behind apertures, the perceived direction of motion is heavily biased by the orientation of the lines/gratings, as well as by the shape and contrast of the apertures [17, 18, 19]. Multiple-aperture stimuli have also recently been used to study coherent motion perception with short-range motion stimulus [20, 21]. For both types of stimuli we compared the model predictions with human performance across various experimental conditions.

## 2 Hierarchical Model for Motion Estimation

Our hierarchical model represents a motion field using a graph $\mathcal{G} = (\mathcal{V}, \mathcal{E})$, which has $L + 1$ hierarchical levels, i.e., $\mathcal{V} = \boldsymbol{\nu}^0 \cup ... \cup \boldsymbol{\nu}^l \cup ... \cup \boldsymbol{\nu}^L$. The level $l$ has a set of nodes $\boldsymbol{\nu}^l = \{\nu^l(i,j), i = 1..., M_l, j = 1..., N_l\}$, forming a 2D lattice indexed by $(i, j)$. More specifically, we start from the pixel lattice and construct the hierarchy as follows.

The nodes $\{\nu^0(i,j)\}$ at the $0^{th}$ level correspond to the pixel position $\{x|x = (i,j)\}$ of the image lattice. We recursively add higher levels with nodes $\boldsymbol{\nu}^l$ ($l = 1, ..., L$). The level $l$ lattice decreases by a factor of 2 along each coordinate direction from level $l - 1$. The edges $\mathcal{E}$ of the graph connect nodes at each level of the hierarchy to nodes in the neighboring levels. Specifically, edges connect node $\nu^l(i,j)$ at level $l$ to a set of child nodes $Ch^l(i,j) = \{\nu^{l-1}(i',j')\}$ at level $l - 1$ satisfying $2i - d \leq i' \leq 2i + d, 2j - d \leq j' \leq 2j + d$. Here $d$ is a parameter controlling how many neighboring nodes in a level share child nodes. Figure 1 illustrates the graph structure of this hierarchical model in the 1-D case and with $d = 2$. Note that our graph $\mathcal{G}$ contains closed loops due to sharing of child nodes.

To apply the model to motion estimation, we define state variable $\mathbf{u}^l(i,j)$ at each node to represent the motion, and connect the $0^{th}$ level nodes to two consecutive image frames, $D = (I_t(x), I_{t+1}(x))$. The problem of motion estimation is to estimate the 2D motion field $\mathbf{u}(x)$ at time $t$ for every pixel site $x$ from input $D$. For simplicity, we use $\mathbf{u}_i^l$ to denote the motion instead of $\mathbf{u}^l(i,j)$ in the following sections.

### 2.1 Model formulation

We define a probability distribution over the motion field $U = \{\mathbf{u}_i^l\}_{l=0}^L$ and $\mathbf{u}^l = \{\mathbf{u}_i^l\}$ on the graph $\mathcal{G}$ conditioned on the input image pair $D$:

$$P(U|D) = \frac{1}{Z} \exp\left(-\left[E_d(D, \mathbf{u}^0) + \sum_{l=0}^{L-1} E_u^l(\mathbf{u}^l, \mathbf{u}^{l+1})\right]\right) \tag{1}$$

where $E_d$ is the data term for the motion based on local image cues and $E_u^l$ are hierarchical priors on the motion which impose slow and smoothness constraints at different levels. Energy terms $E_d, \{E_u^l\}$ are defined using $L_1$ norms to encourage robustness [22]. This robust norm helps deal with the measurement noise that often occur at motion boundary and to prevent over-smoothing at the higher levels. The details of two energy function terms are described as follows:

**1) The Data Term $E_d$**

The data energy term is defined only at the bottom level of the hierarchy. It is specified in terms of the $L_1$ norm between local image intensity values from adjacent frames. More precisely:

$$E_d(D, \mathbf{u}^0) = \sum_i \left(||I_t(x_i) - I_{t+1}(x_i + \mathbf{u}_i^0)||_{L_1} + \alpha||\mathbf{u}_i^0||_{L_1}\right) \tag{2}$$

where the first term defines a difference measure between two measurements centered at $x_i$ in $I_t$ and centered at $x_i + \mathbf{u}_i^0$ in $I_{t+1}$ respectively. We choose to use pixel values only here. The second term imposes a slowness prior on the motion which is weighted by the coefficient $\alpha$. Note that the first term is a matching term that computes the similarity between $I_t(x)$ and $I_{t+1}(x + \mathbf{u})$ given any displacement $\mathbf{u}$. These similarity scores at $x$ gives confidence for different local motion hypotheses: higher similarity means the motion is more likely while lower means it is less likely.

**2) The Hierarchical Prior $\{E_u^l\}$**

We define a hierarchical prior on the slowness and spatial smoothness of motion fields. The first term of this prior is expressed by energy terms between nodes at different levels of the hierarchy and enforces a smoothness preference for their states $\mathbf{u}$ – that the motion of a child node is similar to the motion of its parent. We use the robust $L_1$ norm in the energy terms so that the violation of that consistency constraint will be penalized moderately. This imposes weak smoothness on the motion field and allows abrupt change on motion boundaries. The second term is a $L_1$ norm of motion velocities that encourages the slowness.

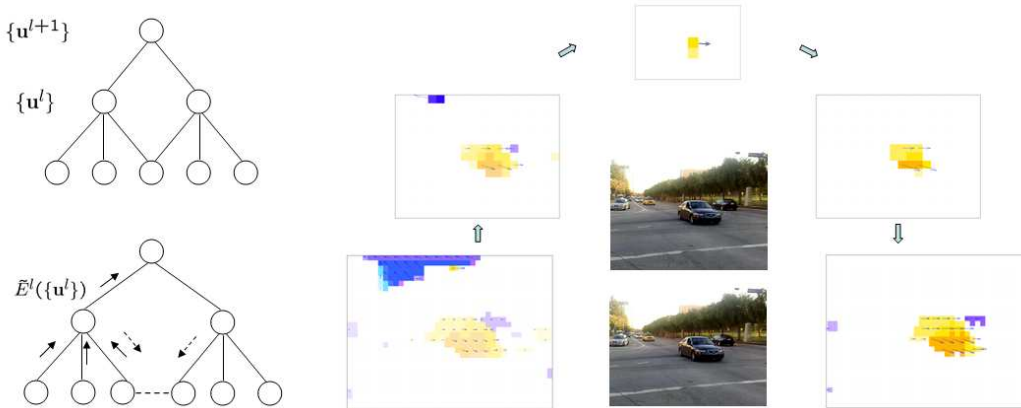

Figure 2: An illustration of our inference procedure. Left top panel: the original hierarchical graph with loops. Left bottom panel: the bottom-up process proceeds on a tree graph with multiple copies of nodes (connected by solid lines) which relaxes the problem. The top-down process enforces the consistency constraints between copies of each node (denoted by dash line connection). Right panel: An example of the inference procedure on two street scene frames. We show the estimates from minimizing $\tilde{E}(U)$ (bottom-up) and $E(U)$ (top-down). The motions are color-coded and also displayed by arrows.

To be specific, the energy function $E_u(\mathbf{u}^l, \mathbf{u}^{l+1})$ is defined to be:

$$E_u^l(\mathbf{u}^l, \mathbf{u}^{l+1}) = \beta(l) \sum_{i \in \boldsymbol{\nu}_{l+1}} \left( \sum_{j \in Ch^{l+1}(i)} ||\mathbf{u}_i^{l+1} - \mathbf{u}_j^l||_{L_1} + \gamma ||\mathbf{u}_i^{l+1}||_{L_1} \right), \qquad (3)$$

where $\beta(l)$ is the weight parameter for the energy terms at the $l^{th}$ level and $\gamma$ controls the relative weight of the slowness prior. Note that our hierarchical smoothness prior differs from conventional smoothness constraints, e.g., [10], because they impose smoothness 'sideways' between neighboring pixels at the same resolution level, which requires that the motion is similar between neighboring sites at the pixel level only. Imposing longer range interactions sideways becomes problematic as it leads to Markov Random Field (MRF) models with a large number of edges. This structure makes it difficult to do inference using standard techniques like belief propagation and max-flow/min-cut. By contrast, we impose smoothness by requiring that child nodes have similar motions to their parent nodes. This 'hierarchical' formulation enables us to impose smoothness interactions at different hierarchy levels while inference can be done efficiently by exploiting the hierarchy.

## 2.2 Motion Estimation

We estimate the motion field by computing the most probable motion $\hat{U} = \arg\max_U P(U|D)$, where $P(U|D)$ was defined as a Gibbs distribution in equation (1). Performing inference on this model is challenging since the energy is defined over a hierarchical graph structure with many closed loops, the state variables $U$ are continuous-valued, and the energy function is non-convex.

Our strategy is to convert this into a discrete optimization problem by quantizing the motion state space. For example, we estimate the motion at an integer-valued resolution if the accuracy is sufficient for certain experimental settings. Given a discrete state space, our algorithm involves bottom-up and top-down processing and is sketched in Figure 2. The algorithm is designed to be parallelizable and to only require computations between neighboring nodes. This is desirable for biological plausibility but also has the practical advantage that we can implement the algorithm using GPU type architectures which enables fast convergence. We describe our inference algorithm in detail as follows.

**i) Bottom-up Pass.** We first approximate the hierarchial graph with a tree-structured model by making multiple copies of child nodes such that each child node has a single parent (see [23]). This enables us to perform exact inference on the relaxed model using dynamic programming. More specifically, we compute an approximate energy function $\tilde{E}(U)$ recursively by exploiting the tree

structure:

$$\tilde{E}(\mathbf{u}_i^{l+1}) = \sum_{j \in Ch^{l+1}(i)} \min_{\mathbf{u}_j^l} [E_u^l(\mathbf{u}_i^{l+1}, \mathbf{u}_j^l) + \tilde{E}(\mathbf{u}_j^l)]$$

where $\tilde{E}(\mathbf{u}_j^0)$ at the bottom level is the data energy $E_d(\mathbf{u}_j^0; D)$. At the top level $L$ we compute the states $(\hat{\mathbf{u}}_i^L)$ which minimize $\tilde{E}(\mathbf{u}_i^L)$.

**ii) Top-down Pass.** Given the top-level motion $(\hat{u}_i^L)$, we then compute the optimal motion configuration for other levels using the following top-down procedure. The top-down pass enforces the consistency constraints, relaxed earlier on the recursively-computed energy function $\tilde{E}$, so that all copies of each node have the same optimal state. We minimize the following energy function recursively for each node:

$$\hat{\mathbf{u}}_j^l = \arg\min_{\mathbf{u}_j^l} [\sum_{i \in Pa^l(j)} E_u^l(\hat{\mathbf{u}}_i^{l+1}; \mathbf{u}_j^l) + \tilde{E}(\mathbf{u}_j^l)]$$

where $Pa^l(j)$ is the set of parents of level-$l$ node $j$. In the top-down pass, the spatial smoothness is imposed to the motion estimates at higher levels which provide context information to disambiguate the motion estimated at lower levels.

The intuition for this two-pass inference algorithm is that the motion estimates of the lower level nodes are typically more ambiguous than the motion estimates of the higher level nodes because the higher levels are able to integrate information from larger number of nodes at lower levels (although some information is lost due to the coarse representation of motion field). Hence the estimates from the higher-level nodes are usually less noisy and can be used to give "context" to resolve the ambiguities of the lower level nodes. From another perspective, this can be thought of as a message-passing type algorithm which uses a specific scheduling scheme [24].

# 3   Experiments with random dot kinematograms

## 3.1   The stimuli and simulation procedures

Random dot kinematogram (RDK) stimuli consist of two image frames with $N$ dots in each frame [1, 16, 6]. As shown in figure (3), the dots in the first frame are located at random positions. A proportion $CN$ of dots (the signal dots) are moved coherently to the second frame with a translational motion. The remaining $(1-C)N$ dots (the noise dots) are moved to random positions in the second frame. The displacement of signal dots are large between the two frames. As a result, the two-frame RDK stimuli are typically considered as an example of long-range motion. The difficulty of perceiving coherent motion in RDK stimuli is due to the large correspondence uncertainty introduced by the noise dots as shown in rightmost panel in figure (3).

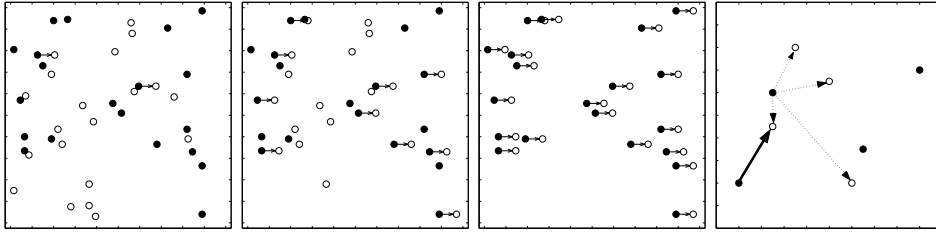

Figure 3: The left three panels show coherent stimuli with $N = 20, C = 0.1$, $N = 20, C = 0.5$ and $N = 20, C = 1.0$ respectively. The closed and open circles denote dots in the first and second frame respectively. The arrows show the motion of those dots which are moving coherently. Correspondence noise is illustrated by the rightmost panel showing that a dot in the first frame has many candidate matches in the second frame.

Barlow and Tripathy [16] used RDK stimuli to investigate how dot density can affect human performance in a global motion discrimination task. They found that human performance (measured by the coherence threshold) vary little with dot density. We tested our model on the same task to judge

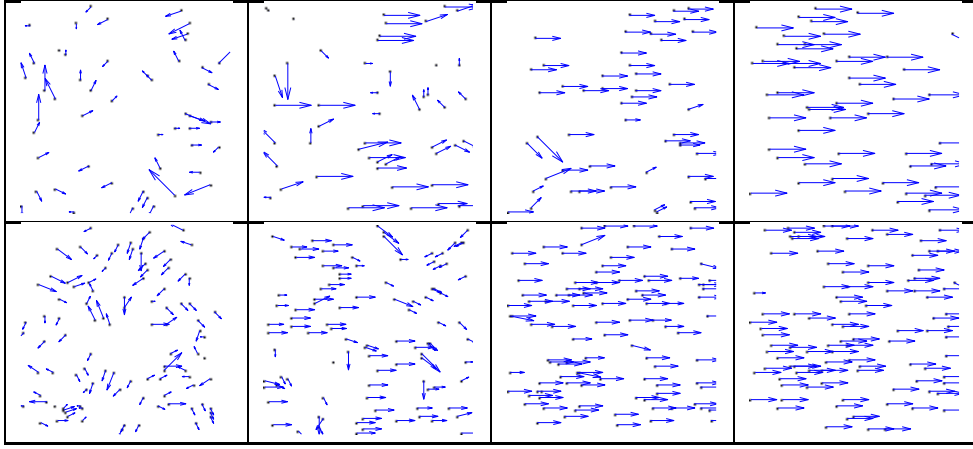

Figure 4: Estimated motion fields for random dot kinematograms. First row: 50 dots in the RDK stimulus; Second row: 100 dots in the RDK stimulus; Column-wise, coherence ratio $C = 0.0, 0.3, 0.6, 0.9$, respectively. The arrows indicate the motion estimated for each dot.

the global motion direction using RDK motion stimulus as the input image. We applied our model to estimate motion fields and used the average velocity to indicate the global motion direction (to the left or to the right). We ran 500 trials for each coherence ratio condition. The dot number varies with $N = 40, 80, 100, 200, 400, 800$ respectively, corresponding to a wide range of dot densities. The model performance was computed for each coherence ratio to fit psychometric functions and to find the coherence threshold at which model performance can reach 75% accuracy.

### 3.2 The Results

Figure (4) shows examples of the estimated motion field for various values of dot number $N$ and coherence ratio $C$. The model outputs provide visually coherent motion estimates when the coherence ratio was greater than 0.3, which is consistent with human perception. With the increase of coherence ratio, the estimated motion flow appears to be more coherent.

To further compare with human performance [16], we examined whether model performance can be affected by dot density in the RDK display. The right plot in figure (5) shows the model performance as a function of the coherence ratio. The coherence threshold, using the criterion of 75% accuracy, showed that model performance varied little with the increase of dot density, which is consistent with human performance reported in psychophysical experiments [16, 6].

## 4 Experiments with multi-aperture stimuli

### 4.1 The two types of stimulus

The multiple-aperture stimulus consisted of a dense set of spatially isolated elements. Two types of elements were used in our simulations: (i) drifting sine-wave gratings with random orientation, and (ii) plaids which includes two gratings with orthogonal orientations. Each element was displayed through a stationary Gaussian window. Figure (6) shows examples of these two types of stimuli.

The grating elements are of form $P_i(\vec{x}, t) = G(\vec{x} - \vec{x}_i, \Sigma) F(\vec{x} - \vec{x}_i - \vec{v}_i t)$ where $\vec{x}_i$ denotes the center of the element, and $F(.)$ represents a grating , $F(x, y) = sin(fx \sin(\theta_i) + fy \cos(\theta_i))$, where $f$ is the fixed spatial frequency and $\theta_i$ is the orientation of the grating.

The grating stimulus is $I(\vec{x}, t) = \sum_{i=1}^{N} P_i(\vec{x}, t)$, where $N$ is the number of elements (which is kept constant). For the $CN$ signal gratings, the motion $\vec{v}_i$ was set to a fixed value $\vec{v}$. For the $(1 - C)N$ noise gratings, we set $|\vec{v}_i| = |\vec{v}|$ and the direction of $\vec{v}_i$ was sampled from a uniform distribution. The grating orientation angles $\theta_i$ were sampled from a uniform distribution also.

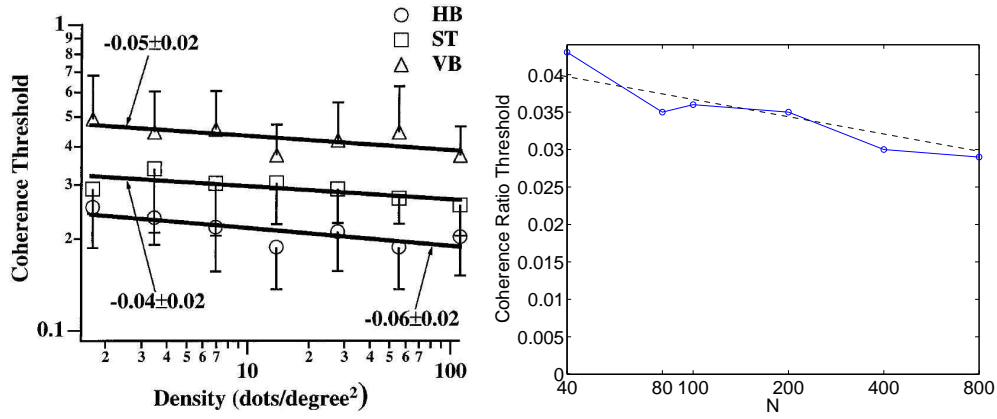

Figure 5: Left panel: Figure 2 in [16] showing that the coherence ratio threshold varies very little with dot density. Right panel: Simulations of our model show a similar trend. $N = 40, 80, 100, 200, 400$ and $800$.

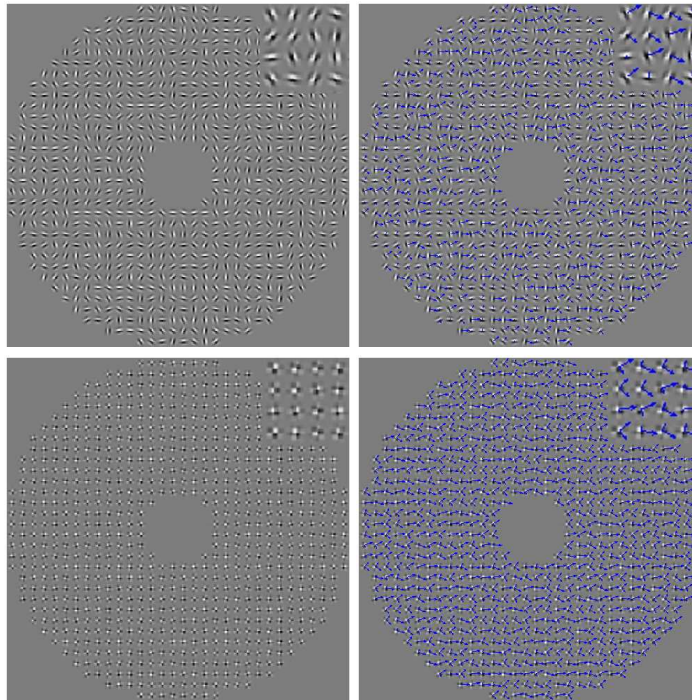

Figure 6: Multi-aperture gratings and plaids. Left column: sample stimuli. Right column: stimuli with the local drifting velocity of each element indicated by arrows. The stimulus details are shown in the magnified windows at the upper right corner of each image.

The plaid elements combine two gratings with orthogonal orientations (each grating has the same speed but can have a different motion direction). This leads to plaid element $Q_i(\vec{x}, t) = G(\vec{x} - \vec{x}_i, \Sigma)\{F_1(\vec{x} - \vec{x}_i - \vec{v}_{i,1}t) + F_2(\vec{x} - \vec{x}_i - \vec{v}_{i,2}t)$, where $F_1(x, y) = sin(fx\sin\theta_i + fy\cos\theta_i)$ and $F_2(x, y) = sin(-fx\cos\theta_i + fy\sin\theta_i)$.

The plaid stimulus is $I(\vec{x}, t) = \sum_{i=1}^{N} Q_i(\vec{x}, t)$. For the $CN$ signal plaids, the motions $\vec{v}_{i,1}, \vec{v}_{i,2}$ were set to a fixed $\vec{v}$. For the $(1 - C)N$ noise plaids, the directions of $\vec{v}_{i,1}, \vec{v}_{i,2}$ were randomly assigned, but their magnitude $|\vec{v}|$ was fixed.

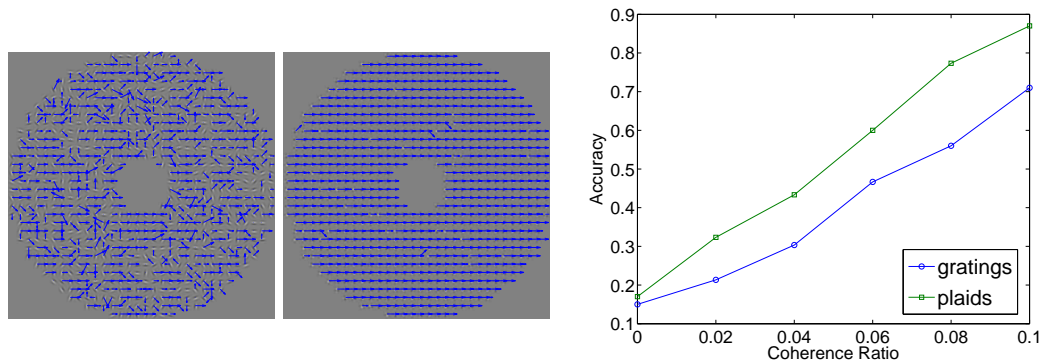

Figure 7: Left two panels: Estimated motion fields of grating and plaids stimuli. Rightmost panel: Psychometric functions of gratings and plaids stimuli.

## 4.2 Simulation procedures and results

The left two panels in Figure (7) show the estimated motion fields for the two types of stimulus we studied with the same coherence ratios 0.7. Plaids stimuli produce more coherent estimated motion field than grating stimuli, which is understandable. because they have less ambiguous local motion cues.

We tested our model in an 8-direction discrimination task for estimating global motion direction [20]. The model used raw images frames as the input. We ran 300 trials for each stimulus type, and used the direction of the average motion to predict the global motion direction. The prediction accuracy – i.e. the number of times our model predicted the correct motion direction from 8 alternatives – was calculated at different coherence ratio levels. This difference between gratings and plaids is shown in the rightmost panel of Figure (7), where the psychometric function of plaids stimuli is always above that of grating stimuli, indicating better performance. These simulation results of our model are consistent with the psychophysics experiments in [20].

## 5 Discussion

In this paper, we proposed a unified single-system framework that is capable of dealing with both short-range and long-range motion. It differs from traditional motion energy models because it does not use spatiotemporal filtering. Note that it was shown in [6] that motion energy models are not well suited to the long-range motion stimuli studied in this paper. The local ambiguities of motion are resolved by a novel hierarchical prior which combines slowness and smoothness at a range of different scales. Our model accounts well for human perception of both short-range and long-range motion using the two standard stimulus types (RDKs and gratings).

The hierarchical structure of our model is partly motivated by known properties of cortical organization. It also has the computational motivation of being able to represent prior knowledge about motion at different scales and to allow efficient computation.

### Acknowledgments

This research was supported by NSF grants IIS-0917141, 613563 to AY and BCS-0843880 to HL. We thank Alan Lee and George Papandreou for helpful discussions.

## References

[1] O. Braddick. A short-range process in apparent motion. *Vision Research*. 14, 519-529. 1974.

[2] Z. Lu, and G. Sperling. Three-systems theory of human visual motion perception: review and update. *Journal of the Optical Society of America. A*. 18, 2331-2369. 2001.

[3]  L. M. Vaina, and S. Soloviev. First-order and second-order motion: neurological evidence for neuroanatomically distinct systems. *Progress in Brain Research*. 144, 197-212. 2004.

[4]  T.S. Lee and D.B. Mumford. Hierarchical Bayesian inference in the visual cortex. *JOSA A*, Vol. 20, Issue 7, pp. 1434-1448. 2003.

[5]  A.L. Yuille and N.M. Grzywacz, A computational theory for the perception of coherent visual motion. *Nature* 333 pp. 71-74. 1988.

[6]  H. Lu and A.L. Yuille. Ideal observers for detecting motion: Correspondence noise. *NIPS*, 2006.

[7]  M. R. Luettgen, W. C. Karl and A. S. Willsky. Efficient Multiscale Regularization with Applications to the Computation of Optical Flow. *IEEE Transactions on image processing*. Vol. 3, pp. 41-64. 1993.

[8]  P. Cavanagh. Short-range vs long-range motion: not a valid distinction. 5(4), pp 303-309. 1991.

[9]  S. Grossberg, and M. E. Rudd. Cortical dynamics of visual motion perception: short-range and long-range apparent motion. *Psychological Review*. 99(1), pp 78-121. 1992.

[10] B.K.P. Horn and B.G. Schunck. Determining Optical Flow. *Artificial Intelligence*. 17(1-3), pp 185-203. 1981.

[11] Y. Weiss, E.P. Simoncelli, and E.H. Adelson. Motion illusions as optimal percepts. *Nature Neuroscience*, 5(6):598-604, Jun 2002.

[12] A. A. Stocker and E. P. Simoncelli. Noise characteristics and prior expectations in human visual speed perception. *Nature Neuroscience*, vol.9(4), pp. 578–585, Apr 2006.

[13] S. Roth and M. J. Black: On the spatial statistics of optical flow. *International Journal of Computer Vision*, 74(1):33-50, August 2007.

[14] P. Anandan. A computational framework and an algorithm for the measurement of visual motion. *Int. Journal. Computer Vision*. 2. pp 283-310. 1989.

[15] K. H. Britten, M. N. Shadlen, W. T. Newsom and J. A. Movshon. The analysis of visual motion: a comparison of neuronal and psychophysical performance. *Journal of Neuroscience*. 12(12), 4745-4765. 1992

[16] H. Barlow, and S.P. Tripathy. Correspondence noise and signal pooling in the detection of coherent visual motion. *Journal of Neuroscience*, 17(20), 7954-7966. 1997.

[17] E. Mingolla, J.T. Todd, and J.F. Norman. The perception of globally coherent motion. *Vision Research*, 32(6), 1015-1031. 1992.

[18] J. Lorenceau, and M. Shiffrar. The influence of terminators on motion integration across space. *Vision Research*, 32(2), 263-273. 1992.

[19] T. Takeuchi. Effect of contrast on the perception of moving multiple Gabor patterns. *Vision research*, 38(20), 3069-3082. 1998.

[20] K. Amano, M. Edwards, D. R. Badcock and S. Nishida. Adaptive pooling of visual motion signals by the human visual system revealed with a novel multi-element stimulus. *Journal of Vision*, 9(3(4)), 1-25. 2009.

[21] A. Lee and H. Lu. A comparison of global motion perception using a multiple-aperture stimulus. *Journal of Vision*. 10(4), 9. 2010.

[22] M. Black and P. Anandan. The robust estimation of multiple motions: Parametric and piecewise-smooth flow fields. *CVIU* 63(1), 1996.

[23] A. Choi, M. Chavira and A. Darwiche. A Scheme for Generating Upper Bounds in Bayesian Networks. *UAI*, 2007.

[24] J. Pearl. *Probabilistic Reasoning in Intelligent Systems: networks of plausible inference*, 1988

